# Multidimensional Scaling and Data Clustering

**Thomas Hofmann  &  Joachim Buhmann**
Rheinische Friedrich–Wilhelms–Universität
Institut für Informatik III, Römerstraße 164
D-53117 Bonn, Germany
email:{th,jb}@cs.uni-bonn.de

## Abstract

Visualizing and structuring pairwise dissimilarity data are difficult combinatorial optimization problems known as *multidimensional scaling* or *pairwise data clustering*. Algorithms for embedding dissimilarity data set in a Euclidian space, for clustering these data and for actively selecting data to support the clustering process are discussed in the maximum entropy framework. Active data selection provides a strategy to discover structure in a data set efficiently with partially unknown data.

## 1  Introduction

Grouping experimental data into compact clusters arises as a data analysis problem in psychology, linguistics, genetics and other experimental sciences. The data which are supposed to be clustered are either given by an explicit coordinate representation (*central* clustering) or, in the *non-metric* case, they are characterized by dissimilarity values for pairs of data points (*pairwise* clustering). In this paper we study algorithms **(i)** for embedding non-metric data in a $D$-dimensional Euclidian space, **(ii)** for simultaneous clustering and embedding of non–metric data, and **(iii)** for active data selection to determine a particular cluster structure with minimal number of data queries. All algorithms are derived from the maximum entropy principle (Hertz *et al.*, 1991) which guarantees robust statistics (Tikochinsky *et al.*, 1984).

The data are given by a real–valued, symmetric proximity matrix $\mathbf{D} \in \mathbf{R}^{N \times N}$, $\mathcal{D}_{kl}$ being the pairwise dissimilarity between the data points $k, l$. Apart from the symmetry constraint we make no further assumptions about the dissimilarities, i.e., we do not require $\mathbf{D}$ being a metric. The numbers $\mathcal{D}_{kl}$ quite often violate the triangular inequality and the dissimilarity of a datum to itself could be finite.

## 2  Statistical Mechanics of Multidimensional Scaling

Embedding dissimilarity data in a $D$-dimensional Euclidian space is a non-convex optimization problem which typically exhibits a large number of local minima. Stochastic search methods like simulated annealing or its deterministic variants have been very successfully

applied to such problems. The question in multidimensional scaling is to find coordinates $\{\mathbf{x}_i\}_{i=1}^N$ in a $D$-dimensional Euclidian space with minimal embedding costs

$$\mathcal{H}^{\mathrm{MDS}} = \frac{1}{2N} \sum_{i,k=1}^N \left[ |\mathbf{x}_i - \mathbf{x}_k|^2 - \mathcal{D}_{ik} \right]^2. \tag{1}$$

Without loss of generality we shift the center of mass in the origin ($\sum_{k=1}^N \mathbf{x}_k = 0$).

In the maximum entropy framework the coordinates $\{\mathbf{x}_i\}$ are regarded as random variables which are distributed according to the Gibbs distribution $P(\{\mathbf{x}_i\}) = \exp(-\beta(\mathcal{H}^{\mathrm{MDS}} - \mathcal{F})$. The inverse temperature $\beta = 1/T$ controls the expected embedding costs $\langle \mathcal{H}^{\mathrm{MDS}} \rangle$ (expectation values are denoted by $\langle . \rangle$). To calculate the free energy $\mathcal{F}$ for $\mathcal{H}^{\mathrm{MDS}}$ we approximate the coupling term $2 \sum_{i,k=1}^N \mathcal{D}_{ik} \mathbf{x}_i \mathbf{x}_k / N \approx \sum_{i=1}^N \mathbf{x}_i \mathbf{h}_i$ with the mean fields $\mathbf{h}_i = 4 \sum_{k=1}^N \mathcal{D}_{ik} \langle \mathbf{x}_k \rangle / N$. Standard techniques to evaluate the free energy $\mathcal{F}$ yield the equations

$$\mathcal{Z}(\mathcal{H}^{\mathrm{MDS}}) \quad \sim \quad \int_{-i\infty}^{i\infty} d\hat{\mathbf{y}} \int_{-\infty}^{\infty} \prod_{d,d'=1}^D d\mathcal{R}_{d,d'} \exp(-\beta N \mathcal{F}), \tag{2}$$

$$\mathcal{F}(\mathcal{H}^{\mathrm{MDS}}) \quad = \quad 2 \sum_{d,d'=1}^D \mathcal{R}_{dd'}^2 - \frac{1}{\beta N} \sum_{i=1}^N \ln \int_{-\infty}^{\infty} d\mathbf{x}_i \exp\left(-\beta f(\mathbf{x}_i)\right), \tag{3}$$

$$f(\mathbf{x}_i) \quad = \quad |\mathbf{x}_i|^4 - \frac{2}{N} |\mathbf{x}_i|^2 \sum_{k=1}^N \mathcal{D}_{ik} + 4\mathbf{x}_i^T \mathcal{R} \mathbf{x}_i + \mathbf{x}_i^T (\mathbf{h}_i - 4\hat{\mathbf{y}}). \tag{4}$$

The integral in Eq. (2) is dominated by the absolute minimum of $\mathcal{F}$ in the limit $N \to \infty$. Therefore, we calculate the saddle point equations

$$\mathcal{R} \quad = \quad \frac{1}{N} \sum_{i=1}^N \left( \langle \mathbf{x}_i \mathbf{x}_i^T \rangle + \frac{1}{2} \langle |\mathbf{x}_i|^2 \rangle \mathcal{I} \right) \quad \text{and} \quad 0 = \frac{1}{N} \sum_{i=1}^N \langle \mathbf{x}_i \rangle \tag{5}$$

$$\langle \mathbf{x}_i \rangle \quad = \quad \frac{\int \mathbf{x}_i \exp(-\beta f(\mathbf{x}_i) d\mathbf{x}_i}{\int \exp(-\beta f(\mathbf{x}_i) d\mathbf{x}_i}. \tag{6}$$

Equation (6) has been derived by differentiating $\mathcal{F}$ with respect to $\mathbf{h}_i$. $\mathcal{I}$ denotes the $D \times D$ unit matrix. In the low temperature limit $\beta \to \infty$ the integral in (3) is dominated by the minimum of $f(\mathbf{x}_i)$. Therefore, a new estimate of $\langle \mathbf{x}_i \rangle$ is calculated minimizing $f$ with respect to $\mathbf{x}_i$. Since all explicit dependencies between the $\mathbf{x}_i$ have been eliminated, this minimization can be performed independently for all i, $1 \le i \le N$.

In the spirit of the EM algorithm for Gaussian mixture models we suggest the following algorithm to calculate a meanfield approximation for the multidimensional scaling problem.

```
initialize ⟨x_i⟩^(0) randomly; t = 0.

while Σ_{i=1}^N |⟨x_i⟩^(t) − ⟨x_i⟩^(t−1)| > ε

    E-step: estimate ⟨x_i⟩^(t+1) as a function of ⟨x_i⟩^(t), R^(t), ŷ^(t), h_i^(t)

    M-step: calculate R^(t), h_i^(t) and determine ŷ^(t) such
            that the centroid condition is satisfied.
```

This algorithm was used to determine the embedding of protein dissimilarity data as shown in Fig. 1d. The phenomenon that the data clusters are arranged in a circular fashion is explained by the lack of small dissimilarity values. The solution in Fig. 1d is about a factor of two better than the embedding found by a classical MDS program (Gower, 1966). This program determines a $(N-1)$- space where the ranking of the dissimilarities is preserved and uses principle component analysis to project this tentative embedding down to two dimensions. Extensions to other MDS cost functions are currently under investigation.

## 3 Multidimensional Scaling and Pairwise Clustering

Embedding data in a Euclidian space precedes quite often a visual inspection by the data analyst to discover structure and to group data into clusters. The question arises how both problems, the embedding problem and the clustering problem, can be solved simultaneously. The second algorithm addresses the problem to embed a data set in a Euclidian space such that the clustering structure is approximated as faithfully as possible in the maximum entropy sense by the clustering solution in this embedding space. The coordinates in the embedding space are the free parameters for this optimization problem.

Clustering of non-metric dissimilarity data, also called pairwise clustering (Buhmann, Hofmann, 1994a), is a combinatorial optimization problem which depends on Boolean assignments $M_{i\nu} \in \{0,1\}$ of datum $i$ to cluster $\nu$. The cost function for pairwise clustering with $K$ clusters is

$$\mathcal{E}_K^{pc}(\mathbf{M}) = \sum_{\nu=1}^{K} \frac{1}{2p_\nu N} \sum_{k=1}^{N} \sum_{l=1}^{N} M_{k\nu} M_{l\nu} \mathcal{D}_{kl} \quad \text{with} \quad p_\nu = \frac{1}{N} \sum_{k=1}^{N} M_{k\nu}. \tag{7}$$

In the meanfield approach we approximate the Gibbs distribution $P(\mathcal{E}_K^{pc})$ corresponding to the original cost function by a family of approximating distributions. The distribution which represents most accurately the statistics of the original problem is determined by the minimum of the Kullback–Leibler divergence to the original Gibbs distribution. In the pairwise clustering case we introduce potentials $\{\mathcal{E}_{k\nu}\}$ for the effective interactions, which define a set of cost functions with non-interacting assignments.

$$\mathcal{E}_K^0(\mathbf{M}, \{\mathcal{E}_{k\nu}\}) = \sum_{\nu=1}^{K} \sum_{k=1}^{N} M_{k\nu} \mathcal{E}_{k\nu}. \tag{8}$$

The optimal potentials derived from this minimization procedure are

$$\{\mathcal{E}_{k\nu}^*\} = \arg \min_{\{\mathcal{E}_{k\nu}\}} \mathcal{D}_{\text{KL}} \left( P^0(\mathcal{E}_K^0) \| P(\mathcal{E}_K^{pc}) \right), \tag{9}$$

where $P^0(\mathcal{E}_K^0)$ is the Gibbs distribution corresponding to $\mathcal{E}_K^0$, and $\mathcal{D}_{\text{KL}}(\cdot\|\cdot)$ is the KL–divergence. This method is equivalent to minimizing an upper bound on the free energy (Buhmann, Hofmann, 1994b),

$$\mathcal{F}(\mathcal{E}_K^{pc}) \leq \mathcal{F}_0(\mathcal{E}_K^0) + \langle \mathcal{V}_K \rangle_0, \quad \text{with} \quad \mathcal{V}_K = \mathcal{E}_K^{pc} - \mathcal{E}_K^0, \tag{10}$$

$\langle . \rangle_0$ denoting the average over all configurations of the cost function without interactions.

Correlations between assignment variables are statistically independent for $P^0(\mathcal{E}_K^0)$, i.e., $\langle M_{k\nu} M_{l\nu} \rangle_0 = \langle M_{k\nu} \rangle_0 \langle M_{l\nu} \rangle_0$. The averaged potential $\mathcal{V}_K$, therefore, amounts to

$$\langle \mathcal{V}_K \rangle = \sum_{\nu=1}^{K} \sum_{k,l=1}^{N} \langle M_{k\nu} \rangle \langle M_{l\nu} \rangle \frac{1}{2p_\nu N} \mathcal{D}_{kl} - \sum_{\nu=1}^{K} \sum_{k=1}^{N} \langle M_{k\nu} \rangle \mathcal{E}_{k\nu}, \tag{11}$$

the subscript of averages being omitted for conciseness. The expected assignment variables are

$$\langle M_{i\nu} \rangle = \frac{\exp(-\beta \mathcal{E}_{i\nu})}{\sum_{\mu=1}^{K} \exp(-\beta \mathcal{E}_{i\mu})}. \tag{12}$$

Minimizing the upper bound yields

$$\frac{\partial}{\partial \mathcal{E}_{i\alpha}} \left[ \mathcal{F}_0(\mathcal{E}_K^0) + \langle \mathcal{V}_K \rangle \right] = - \sum_{\nu=1}^{K} \frac{\partial \langle M_{i\nu} \rangle}{\partial \mathcal{E}_{i\alpha}} \left( \mathcal{E}_{i\nu} - \mathcal{E}_{i\nu}^* \right) = 0. \tag{13}$$

The "optimal" potentials

$$\mathcal{E}_{i\nu}^* = \frac{1}{p_\nu N} \sum_{k=1}^{N} \langle M_{k\nu} \rangle \left( \mathcal{D}_{ik} - \frac{1}{2 p_\nu N} \sum_{l=1}^{N} \langle M_{l\nu} \rangle \mathcal{D}_{kl} \right) \tag{14}$$

depend on the given distance matrix, the averaged assignment variables and the cluster probabilities. They are optimal in the sense, that if we set

$$\mathcal{E}_{i\nu} = \mathcal{E}_{i\nu}^* + c_i \tag{15}$$

the $N*K$ stationarity conditions (13) are fulfilled for every $i \in \{1,...,N\}, \nu \in \{1,...,K\}$. A simultaneous solution of Eq. (15) with (12) constitutes a necessary condition for a minimum of the upper bound for the free energy $\mathcal{F}$.

The connection between the clustering and the multidimensional scaling problem is established, if we restrict the potentials $\mathcal{E}_{i\nu}$ to be of the form $|\mathbf{x}_i - \mathbf{y}_\nu|^2$ with the centroids $\mathbf{y}_\nu = \sum_{k=1}^{N} M_{k\nu} \mathbf{x}_\nu / \sum_{k=1}^{N} M_{k\nu}$. We consider the coordinates $\mathbf{x}_i$ as the variational parameters. The additional constraints restrict the family of approximating distributions, defined by $\mathcal{E}_K^0$ to a subset. Using the chain rule we can calculate the derivatives of the upper bound (10), resulting in the exact stationary conditions for $\mathbf{x}_i$,

$$\sum_{\alpha,\nu=1}^{K} \langle M_{i\alpha} \rangle \langle M_{j\alpha} \rangle (\Delta \mathcal{E}_{i\alpha} - \Delta \mathcal{E}_{i\nu}) \mathbf{y}_\alpha = \sum_{j=1}^{N} \sum_{\alpha,\nu=1}^{K} \frac{\langle M_{j\alpha} \rangle \langle M_{j\nu} \rangle}{N p_\alpha} \times$$
$$(\Delta \mathcal{E}_{i\alpha} - \Delta \mathcal{E}_{i\nu}) \left[ \langle M_{i\alpha} \rangle I + \sum_{k=1}^{N} \left( (\mathbf{x}_k - \mathbf{y}_\alpha) \frac{\partial \langle M_{k\alpha} \rangle}{\partial \mathbf{x}_i}^T \right) \right] (\mathbf{x}_j - \mathbf{y}_\alpha), \tag{16}$$

where $\Delta \mathcal{E}_{i\alpha} = \mathcal{E}_{i\alpha} - \mathcal{E}_{i\alpha}^*$. The derivatives $\partial \langle M_{k\alpha} \rangle / \partial \mathbf{x}_i$ can be exactly calculated, since they are given as the solutions of an linear equation system with $N \times K$ unknowns for every $\mathbf{x}_i$. To reduce the computational complexity an approximation can be derived under the assumption $\partial \mathbf{y}_\alpha / \partial \mathbf{x}_i \approx 0$. In this case the right hand side of (16) can be set to zero in a first order approximation yielding an explicit formula for $\mathbf{x}_i$,

$$\mathcal{K}_i \mathbf{x}_i \approx \frac{1}{2} \sum_{\nu=1}^{K} \langle M_{i\nu} \rangle \left( \|\mathbf{y}_\nu\|^2 - \mathcal{E}_{i\nu}^* \right) \left( \mathbf{y}_\nu - \sum_{\alpha=1}^{K} \langle M_{i\alpha} \rangle \mathbf{y}_\alpha \right), \tag{17}$$

with the covariance matrix $\mathcal{K}_i = (\langle \mathbf{y}\mathbf{y}^T \rangle_i - \langle \mathbf{y} \rangle_i \langle \mathbf{y} \rangle_i^T)$ and $\langle \mathbf{y} \rangle_i = \sum_{\nu=1}^{K} \langle M_{i\nu} \rangle \mathbf{y}_\nu$.

The derived system of transcendental equations given by (12), (17) and the centroid condition explicitly reflects the dependencies between the clustering procedure and the Euclidian representation. Solving these equations simultaneously leads to an efficient algorithm which

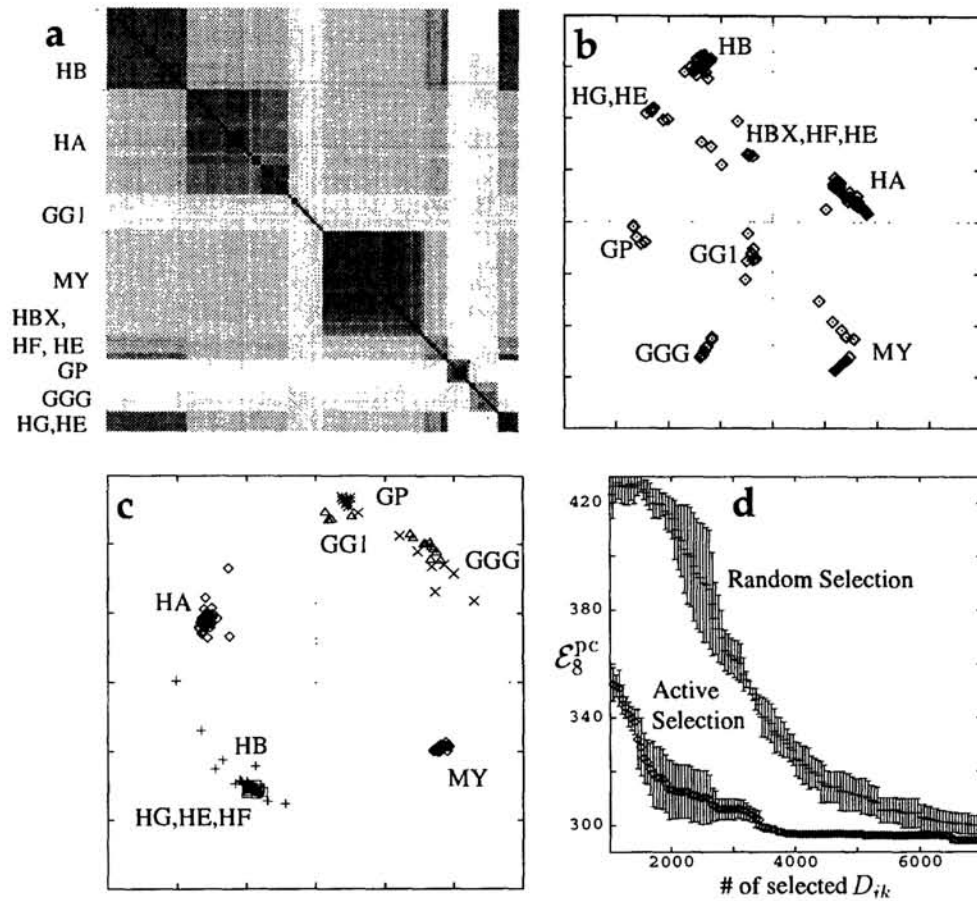

Figure 1: Similarity matrix of 145 protein sequences of the globin family (a): dark gray levels correspond to high similarity values; (b): clustering with embedding in two dimensions; (c): multidimensional scaling solution for 2-dimensional embedding; (d): quality of clustering solution with random and active data selection of $\mathcal{D}_{ik}$ values. $\mathcal{E}_8^{pc}$ has been calculated on the basis of the complete set of $\mathcal{D}_{ik}$ values.

interleaves the multidimensional scaling process and the clustering process and which avoids an artificial separation into two uncorrelated processes. The described algorithm for simultaneous Euclidian embedding and data clustering can be used for dimensionality reduction, e.g., high dimensional data can be projected to a low dimensional subspace in a nonlinear fashion which resembles local principle component analysis (Buhmann, Hofmann, 1994b).

Figure (1) shows the clustering result for a real–world data set of 145 protein sequences. The similarity values between pairs of sequences are determined by a sequence alignment program which takes biochemical and structural information into account. The sequences belong to different protein families like hemoglobin, myoglobin and other globins; they are abbreviated with the displayed capital letters. The gray level visualization of the dissimilarity matrix with dark values for similar protein sequences shows the formation of distinct "squares" along the main diagonal. These squares correspond to the discovered partition after clustering. The embedding in two dimensions shows inter–cluster distances which are in consistent agreement with the similarity values of the data. In three and four dimensions the error between the

given dissimilarities and the constructed distances is further reduced. The results are in good agreement with the biological classification.

## 4   Active Data Selection for Data Clustering

Active data selection is an important issue for the analysis of data which are characterized by pairwise dissimilarity values. The size of the distance matrix grows like the square of the number of data 'points'. Such a $\mathcal{O}(N^2)$ scaling renders the data acquisition process expensive. It is, therefore, desirable to couple the data analysis process to the data acquisition process, i.e., to actively query the supposedly most relevant dissimilarity values. Before addressing active data selection questions for data clustering we have to discuss the problem how to modify the algorithm in the case of incomplete data.

If we want to avoid any assumptions about statistical dependencies, it is impossible to infer unknown values and we have to work directly with the partial dissimilarity matrix. Since the data enters only in the (re-)calculation of the potentials in (14), it is straightforward to appropriately modify these equations. All sums are restricted to terms with known dissimilarities and the normalization factors are adjusted accordingly.

Alternatively we can try to explicitly estimate the unknown dissimilarity values based on a statistical model. For this purpose we propose two models, relying on a known group structure of the data. The first model (I) assumes that all dissimilarities between a point $i$ and points $j$ belonging to a group $G_\mu$ are i.i.d. random variables with the probability density $p_{i\mu}$ parameterized by $\theta_{i\mu}$. In this scheme a subset of the known dissimilarities of $i$ and $j$ to other points $k$ are used as samples for the estimation of $\mathcal{D}_{ij}$. The selection of the specific subset is determined by the clustering structure. In the second model (II) we assume that the dissimilarities between groups $G_\nu, G_\mu$ are i.i.d. random variables with density $p_{\nu\mu}$ parameterized by $\theta_{\nu\mu}$. The parameters $\theta_{\nu\mu}$ are estimated on the basis of all known dissimilarities $\{\mathcal{D}_{ij} \in \mathcal{D}\}$ between points from $G_\nu$ and $G_\mu$.

The assignments of points to clusters are not known a priori and have to be determined in the light of the (given and estimated) data. The data selection strategy becomes self-consistent if we interpret the mean fields $\langle M_{i\nu} \rangle$ of the clustering solution as posterior probabilities for the binary assignment variables. Combined with a maximum likelihood estimation for the unknown parameters given the posteriors, we arrive at an EM–like iteration scheme with the E–step replaced by the clustering algorithm.

The precise form of the M–Step depends on the parametric form of the densities $p_{i\mu}$ or $p_{\nu\mu}$, respectively. In the case of Gaussian distributions the M–Step is described by the following estimation equations for the location parameters

$$\bar{m}_{i\mu}^{(l)} = \frac{\sum_{\mathcal{D}_{ij} \in \mathcal{D}} \langle M_{j\mu} \rangle \mathcal{D}_{ij}}{\sum_{\mathcal{D}_{ij} \in \mathcal{D}} \langle M_{j\mu} \rangle} \quad \text{(I)}, \qquad \bar{m}_{\nu\mu}^{(l)} = \frac{\sum_{\mathcal{D}_{ij} \in \mathcal{D}} \pi_{\nu\mu}^{ij} \mathcal{D}_{ij}}{\sum_{\mathcal{D}_{ij} \in \mathcal{D}} \pi_{\nu\mu}^{ij}} \quad \text{(II)}, \qquad (18)$$

with $\pi_{\nu\mu}^{ij} = \frac{1}{1+\delta_{\nu\mu}} \left( \langle M_{i\nu} \rangle \langle M_{j\mu} \rangle + \langle M_{i\mu} \rangle \langle M_{j\nu} \rangle \right)$. Corresponding expressions are derived for the standard deviations $\bar{\sigma}_{i\nu}^{(l)}$ or $\bar{\sigma}_{\nu\mu}^{(l)}$, respectively. In the case of non–normal distributions the empirical mean might still be a good estimator of the location parameter, though not necessarily a maximum likelihood estimator. The missing dissimilarities are estimated by the following statistics, derived from the empirical means.

$$\bar{D}_{ij}^{(l)} = \sum_{\nu,\mu=1}^{K} \langle M_{i\nu} \rangle \langle M_{j\mu} \rangle \frac{N_{i\mu} \bar{m}_{i\mu}^{(l)} + N_{j\nu} \bar{m}_{j\nu}^{(l)}}{N_{i\mu} + N_{j\nu}} \quad \text{(I)}, \qquad \bar{D}_{ij}^{(l)} = \sum_{\nu \le \mu} \pi_{\nu\mu}^{ij} \bar{m}_{\nu\mu}^{(l)} \quad \text{(II)}, \qquad (19)$$

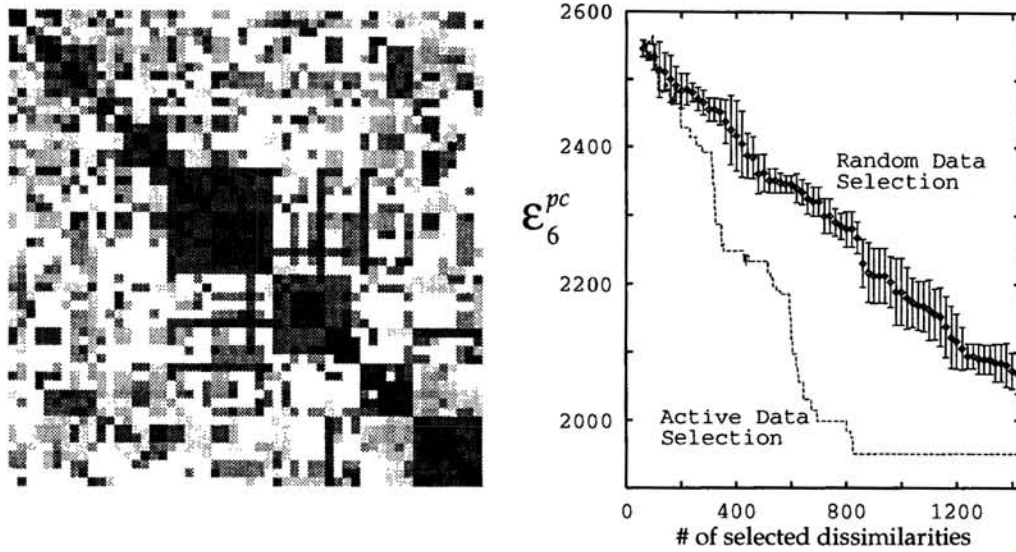

Figure 2: Similarity matrix of 54 word fragments generated by a dynamic programming algorithm. The clustering costs in the experiment with active data selection requires only half as much data as a random selection strategy.

with $N_{i\nu} = \sum_{\mathcal{D}_{ik} \in \mathcal{D}} \langle M_{k\nu} \rangle$. For model (I) we have used a pooled estimator to exploit the data symmetry. The iteration scheme finally leads to estimates $\bar{\theta}_{i\mu}$ or $\bar{\theta}_{\nu\mu}$ respectively for the parameters and $\bar{D}_{ij}$ for all unknown dissimilarities.

**Criterion for Active Data Selection**: We will use the expected reduction in the variance of the free energy $\mathcal{F}_0$ as a score, which should be maximized by the selection criterion. $\mathcal{F}_0$ is given by $\mathcal{F}_0(D) = -\frac{1}{\beta} \sum_{i=1}^{N} \log \sum_{\nu=1}^{K} \exp(-\beta \mathcal{E}_{i\nu}(D))$. If we query a new dissimilarity $\mathcal{D}_{ij}$ the expected reduction of the variance of the free energy is approximated by

$$\Delta_{ij} = 2 \left[ \frac{\partial \mathcal{F}_0}{\partial \mathcal{D}_{ij}} \right]^2 \mathbf{V} \left[ \mathcal{D}_{ij} - \bar{D}_{ij} \right] \tag{20}$$

The partial derivatives can be calculated exactly by solving a system of linear equations with $N \times K$ unknowns. Alternatively a first order approximation in $\epsilon_\nu = \mathcal{O}(1/Np_\nu)$ yields

$$\frac{\partial \mathcal{F}_0}{\partial \mathcal{D}_{ij}} \approx 2 \sum_{\nu=1}^{K} \frac{\langle M_{i\nu} \rangle \langle M_{j\nu} \rangle}{Np_\nu} + O\left( \sum_{\nu=1}^{K} \epsilon_\nu^2 \right). \tag{21}$$

This expression defines a relevance measure of $\mathcal{D}_{ij}$ for the clustering problem since a $\mathcal{D}_{ij}$ value contributes to the clustering costs only if the data $i$ and $j$ belong to the same cluster. Equation (21) summarizes the mean–field contributions $\partial \mathcal{F}_0 / \partial \mathcal{D}_{ij} \approx \partial \langle H \rangle_0 / \partial \mathcal{D}_{ij}$.

To derive the final form of our scoring function we have to calculate an approximation of the variance in Eq. (20) which measures the expected squared error for replacing the true value $\mathcal{D}_{ij}$ with our estimate $\bar{D}_{ij}$. Since we assumed statistical independence the variances are additive $\mathbf{V} \left[ \mathcal{D}_{ij} - \bar{D}_{ij} \right] = \mathbf{V} \left[ \mathcal{D}_{ij} \right] + \mathbf{V} \left[ \bar{D}_{ij} \right]$. The total population variance is a sum of inner- and inter-cluster variances, that can be approximated by the empirical means and by the empirical variances instead of the unknown parameters of $p_{i\nu}$ or $p_{\nu\mu}$. The sampling variance of the statistics $\bar{D}_{ij}$ is estimated under the assumption, that the empirical means $\bar{m}_{i\nu}$

or $\bar{m}_{\nu\mu}$ respectively are uncorrelated. This holds in the hard clustering limit. We arrive at the following final expression for the variances of model (II)

$$\mathbf{V}\left[\mathcal{D}_{ij}-\bar{D}_{ij}\right] \approx \sum_{\nu\leq\mu}\pi_{\nu\mu}^{ij}\left[(\bar{D}_{ij}-\bar{m}_{\nu\mu})^2+\left(1+\frac{\pi_{\nu\mu}^{ij}}{\sum_{\mathcal{D}_{kl}\in\mathcal{D}}\pi_{\nu\mu}^{kl}}\bar{\sigma}_{\nu\mu}^2\right)\right] \quad (22)$$

For model (I) a slightly more complicated formula can be derived. Inserting the estimated variances into Eq. (20) leads to the final expression for our scoring function.

To demonstrate the efficiency of the proposed selection strategy, we have compared the clustering costs achieved by active data selection with the clustering costs resulting from randomly queried data. Assignments int the case of active selection are calculated with statistical model (I). Figure 1d demonstrates that the clustering costs decrease significantly faster when the selection criterion (20) is implemented. The structure of the clustering solution has been completely inferred with about 3300 selected $\mathcal{D}_{ik}$ values. The random strategy requires about 6500 queries for the same quality. Analogous comparison results for linguistic data are summarized in Fig. 2. Note the inconsistencies in this data set reflected by small $\mathcal{D}_{ik}$ values outside the cluster blocks (dark pixels) or by the large $\mathcal{D}_{ik}$ values (white pixels) inside a block.

**Conclusion**: Data analysis of dissimilarity data is a challenging problem in molecular biology, linguistics, psychology and, in general, in pattern recognition. We have presented three strategies to visualize data structures and to inquire the data structure by an efficient data selection procedure. The respective algorithms are derived in the maximum entropy framework for maximal robustness of cluster estimation and data embedding. Active data selection has been shown to require only half as much data for estimating a clustering solution of fixed quality compared to a random selection strategy. We expect the proposed selection strategy to facilitate maintenance of genome and protein data bases and to yield more robust data prototypes for efficient search and data base mining.

**Acknowledgement**: It is a pleasure to thank M. Vingron and D. Bavelier for providing the protein data and the linguistic data, respectively. We are also grateful to A. Polzer and H.J. Warneboldt for implementing the MDS algorithm. This work was partially supported by the Ministry of Science and Research of the state Nordrhein-Westfalen.

# References

Buhmann, J., Hofmann, T. (1994a). Central and Pairwise Data Clustering by Competitive Neural Networks. *Pages 104–111 of: Advances in Neural Information Processing Systems 6*. Morgan Kaufmann Publishers.

Buhmann, J., Hofmann, T. (1994b). A Maximum Entropy Approach to Pairwise Data Clustering. *Pages 207–212 of: Proceedings of the International Conference on Pattern Recognition, Hebrew University, Jerusalem*, vol. II. IEEE Computer Society Press.

Gower, J. C. (1966). Some distance properties of latent root and vector methods used in multivariate analysis. *Biometrika*, **53**, 325–328.

Hertz, J., Krogh, A., Palmer, R. G. (1991). *Introduction to the Theory of Neural Computation*. New York: Addison Wesley.

Tikochinsky, Y., Tishby, N.Z., Levine, R. D. (1984). Alternative Approach to Maximum–Entropy Inference. *Physical Review A*, **30**, 2638–2644.